# Serial Order in Reading Aloud: Connectionist Models and Neighborhood Structure

**Jeanne C. Milostan**
Computer Science & Engineering 0114
University of California San Diego
La Jolla, CA 92093-0114

**Garrison W. Cottrell**
Computer Science & Engineering 0114
University of California San Diego
La Jolla, CA 92093-0114

## Abstract

Dual-Route and Connectionist Single-Route models of reading have been at odds over claims as to the correct explanation of the reading process. Recent Dual-Route models predict that subjects should show an increased naming latency for irregular words when the irregularity is earlier in the word (e.g. chef is slower than glow) - a prediction that has been confirmed in human experiments. Since this would appear to be an effect of the left-to-right reading process, Coltheart & Rastle (1994) claim that Single-Route parallel connectionist models cannot account for it. A refutation of this claim is presented here, consisting of network models which do show the interaction, along with orthographic neighborhood statistics that explain the effect.

## 1 Introduction

A major component of the task of learning to read is the development of a mapping from orthography to phonology. In a complete model of reading, message understanding must play a role, but many psycholinguistic phenomena can be explained in the context of this simple mapping task. A difficulty in learning this mapping is that in a language such as English, the mapping is *quasiregular* (Plaut et al., 1996); there are a wide range of exceptions to the general rules. As with nearly all psychological phenomena, more frequent stimuli are processed faster, leading to shorter naming latencies. The regularity of mapping interacts with this variable, a robust finding that is well-explained by connectionist accounts (Seidenberg and McClelland, 1989; Taraban and McClelland, 1987).

In this paper we consider a recent effect that seems difficult to account for in terms of the standard parallel network models. Coltheart & Rastle (1994) have shown

| Filler | | **Position** 1 | **of** 2 | **Irregular** 3 | **Phoneme** 4 | 5 |
|---|---|---|---|---|---|---|
| Nonword | | | | | | |
| | Irregular | 554 | 542 | 530 | 529 | 537 |
| | Regular Control | 502 | 516 | 518 | 523 | 525 |
| | Difference | 52 | 26 | 12 | 6 | 12 |
| | | | | | | |
| Exception | | | | | | |
| | Irregular | 545 | 524 | 528 | 526 | 528 |
| | Regular Control | 500 | 503 | 503 | 515 | 524 |
| | Difference | 45 | 21 | 25 | 11 | 4 |
| | Avg. Diff. | 48.5 | 23.5 | 18.5 | 8.5 | 8 |

Table 1: Naming Latency vs. Irregularity Position

that the amount of delay experienced in naming an exception word is related to the phonemic position of the irregularity in pronunciation. Specifically, the earlier the exception occurs in the word, the longer the latency to the onset of pronouncing the word. Table 1, adapted from (Coltheart and Rastle, 1994) shows the response latencies to two-syllable words by normal subjects. There is a clear left-to-right ranking of the latencies compared to controls in the last row of the Table. Coltheart *et al.* claim this delay ranking cannot be achieved by standard connectionist models. This paper shows this claim to be false, and shows that the origin of the effect lies in a statistical regularity of English, related to the number of "friends" and "enemies" of the pronunciation within the word's neighborhood [1].

## 2   Background

Computational modeling of the reading task has been approached from a number of different perspectives. Advocates of a dual-route model of oral reading claim that two separate routes, one lexical (a lexicon, often hypothesized to be an associative network) and one rule-based, are *required* to account for certain phenomena in reaction times and nonword pronunciation seen in human subjects (Coltheart et al., 1993). Connectionist modelers claim that the same phenomena can be captured in a single-route model which learns simply by exposure to a representative dataset (Seidenberg and McClelland, 1989).

In the Dual-Route Cascade model (DRC) (Coltheart et al., 1993), the lexical route is implemented as an Interactive Activation (McClelland and Rumelhart, 1981) system, while the non-lexical route is implemented by a set of grapheme-phoneme correspondence (GPC) rules learned from a dataset. Input at the letter identification layer is activated in a left-to-right sequential fashion to simulate the reading direction of English, and fed simultaneously to the two pathways in the model. Activation from both the GPC route and the lexicon route then begins to interact at the output (phoneme) level, starting with the phonemes at the beginning of the word. If the GPC and the lexicon agree on pronunciation, the correct phonemes will be activated quickly. For words with irregular pronunciation, the lexicon and GPC routes will activate different phonemes: the GPC route will try to activate the regular pronunciation while the lexical route will activate the irregular (correct)

pronunciation. Inhibitory links between alternate phoneme pronunciations will slow down the rise in activation, causing words with inconsistencies to be pronounced more slowly than regular words. This slowing will not occur, however, when an irregularity appears late in a word. This is because in the model the lexical node spreads activation to *all* of a word's phonemes as soon as it becomes active. If an irregularity is late in a word, the correct pronunciation will begin to be activated before the GPC route is able to vote against it. Hence late irregularities will not be as affected by conflicting information. This result is validated by simulations with the one-syllable DRC model (Coltheart and Rastle, 1994).

Several connectionist systems have been developed to model the orthography to phonology process (Seidenberg and McClelland, 1989; Plaut et al., 1996). These connectionist models provide evidence that the task, with accompanying phenomena, can be learned through a single mechanism. In particular, Plaut *et al.* (henceforth PMSP) develop a recurrent network which duplicates the naming latencies appropriate to their data set, consisting of approximately 3000 one-syllable English words (monosyllabic words with frequency greater than 1 in the Kucera & Francis corpus (Kucera and Francis, 1967)). Naming latencies are computed based on time-to-settle for the recurrent network, and based on MSE for a feed-forward model used in some simulations. In addition to duplicating frequency and regularity interactions displayed in previous human studies, this model also performs appropriately in providing pronunciation of pronounceable nonwords. This provides an improvement over, and a validation of, previous work with a strictly feed-forward network (Seidenberg and McClelland, 1989). However, to date, no one has shown that Coltheart's naming latency by irregularity of position interaction can be accounted for by such a model. Indeed, it is difficult to see how such a model *could* account for such a phenomenon, as its explanation (at least in the DRC model) seems to require the serial, left-to-right nature of processing in the model, whereas networks such as PMSP present the word orthography all at once. In the following, we fill this gap in the literature, and explain why a parallel, feed-forward model *can* account for this result.

## 3 Experiments & Results

### 3.1 The Data

Pronunciations for approximately 100,000 English words were obtained through an electronic dictionary developed by CMU [2]. The provided format was not amenable to an automated method for distinguishing the number of syllables in the word. To obtain syllable counts, English two-syllable words were gathered from the Medical Research Council (MRC) Psycholinguistic Database (Coltheart and Rastle, 1994), which is conveniently annotated with syllable counts and frequency (only those with Kucera-Francis written frequency of one or greater were selected). Intersecting the two databases resulted in 5,924 two-syllable words. There is some noise in the data; ZONED and AERIAL, for example, are in this database of purported two-syllable words. Due to the size of the database and time limitations, we did not prune the data of these errors, nor did we eliminate proper nouns or foreign words. Single-syllable words with the same frequency criterion were also selected for comparison with previous work. 3,284 unique single-syllable words were obtained, in contrast to 2,998 words used by PMSP. Similar noisy data as in the two-syllable set exists in this database. Each word was represented using the orthography and phonology representation scheme outlined by PMSP.

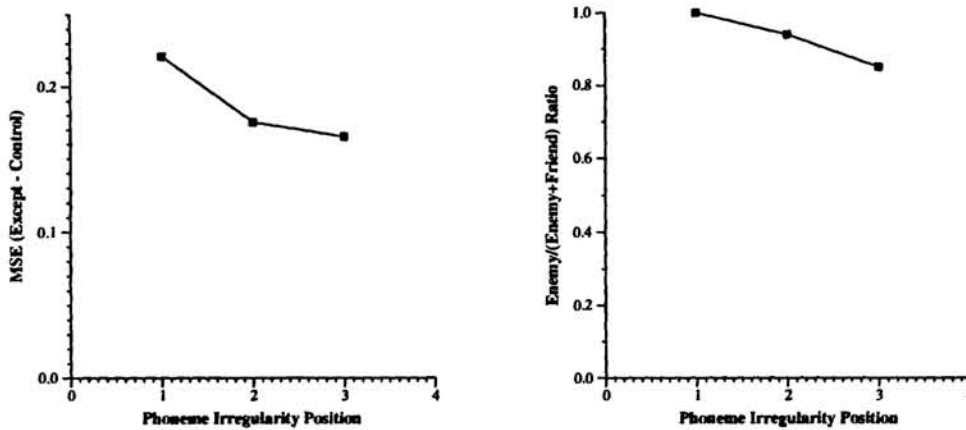

Figure 1: 1-syllable network latency differences & neighborhood statistics

## 3.2   Methods

For the single syllable words, we used an identical network to the feed-forward network used by PMSP, i.e., a 105-100-61 network, and for the two syllable words, we simply used the same architecture with the each layer size doubled. We trained each network for 300 epochs, using batch training with a cross entropy objective function, an initial learning rate of 0.001, momentum of 0.9 after the first 10 epochs, weight decay of 0.0001, and delta-bar-delta learning rate adjustment. Training exemplars were weighted by the log of frequency as found in the Kucera-Francis corpus. After this training, the single syllable feed-forward networks averaged 98.6% correct outputs, using the same evaluation technique outlined in PMSP. Two syllable networks were trained for 1700 epochs using online training, a learning rate of 0.05, momentum of 0.9 after the first 10 epochs, and raw frequency weighting. The two syllable network achieved 85% correct. Naming latency was equated with network output MSE; for successful results, the error difference between the irregular words and associated control words should decrease with irregularity position.

## 3.3   Results

**Single Syllable Words**   First, Coltheart's challenge that a single-route model cannot produce the latency effects was explored. The single-syllable network described above was tested on the collection of single-syllable words identified as irregular by (Taraban and McClelland, 1987). In (Coltheart and Rastle, 1994), control words are selected based on equal number of letters, same beginning phoneme, and Kucera-Francis frequency between 1 and 20 (controls were not frequency matched). For single syllable words used here, the control condition was modified to allow frequency from 1 to 70, which is the range of the "low frequency" exception words in the Taraban & McClelland set. Controls were chosen by drawing randomly from the words meeting the control criteria.

Each test and control word input vector was presented to the network, and the MSE at the output layer (compared to the expected correct target) was calculated. From these values, the differences in MSE for target and matched control words were calculated and are shown in Figure 1. Note that words with an irregularity in the first phoneme position have the largest difference from their control words, with this (exception - regular control) difference decreasing as phoneme position increases. Contrary to the claims of the Dual-Route model, this network does show the desired rank-ordering of MSE/latency.

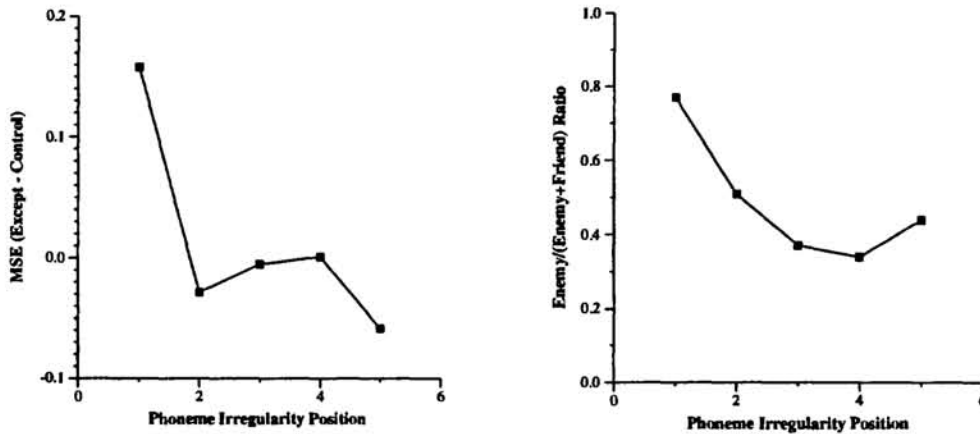

Figure 2: 2-syllable network latency differences & neighborhood statistics

**Two Syllable Words**  Testing of the two-syllable network is identical to that of the one-syllable network. The difference in MSE for each test word and its corresponding control is calculated, averaging across all test pairs in the position set. Both test words and their controls are those found in (Coltheart and Rastle, 1994). The 2-syllable network appears to produce approximately the correct linear trend in the naming MSE/latency (Figure 2), although the results displayed are not monotonically decreasing with position. Note, however, that the results presented by Coltheart, when taken separately, also fail to exhibit this trend (Table 1). For correct analysis, several "subject" networks should be trained, with formal linear trend analysis then performed with the resulting data. These further simulations are currently being undertaken.

## 4   Why the network works: Neighborhood effects

A possible explanation for these results relies on the fact that connectionist networks tend to extract statistical regularities in the data, and are affected by regularity by frequency interactions. In this case, we decided to explore the hypothesis that the results could be explained by a *neighborhood effect*: Perhaps the number of "friends" and "enemies" in the neighborhood (in a sense to be defined below) of the exception word varies in English in a position-dependent way. If there are more enemies (different pronunciations) than friends (identical pronunciations) when the exception occurs at the beginning of a word than at the end, then one would expect a network to reflect this statistical regularity in its output errors. In particular, one would expect higher errors (and therefore longer latencies in naming) if the word has a higher proportion of enemies in the neighborhood.

To test this hypothesis, we created some data search engines to collect word neighborhoods based on various criteria. There is no consensus on the exact definition of the "neighborhood" of a word. There are some common measures, however, so we explored several of these. Taraban & McClelland (1987) neighborhoods (T&M) are defined as words containing the same vowel grouping and final consonant cluster. These neighborhoods therefore tend to consist of words that rhyme (MUST, DUST, TRUST). There is independent evidence that these word-body neighbors are psychologically relevant for word naming tasks (i.e., pronunciation) (Treiman and Chafetz, 1987). The neighborhood measure given by Coltheart (Coltheart and Rastle, 1994), $N$, counts same-length words which differ by only one letter, taking string position into account. Finally, edit-distance-1 (ED1) neighborhoods are those words which can be generated from the target word by making one change

(Peereman, 1995): either a letter substitution, insertion or deletion. This differs from the Coltheart $N$ definition in that "TRUST" is in the ED1 neighborhood (but not the $N$ neighborhood) of "RUST", and provides a neighborhood measure which considers both pronunciation and spelling similarity. However, the $N$ and the ED-1 measure have not been shown to be psychologically real in terms of affecting naming latency (Treiman and Chafetz, 1987).

We therefore extended T&M neighborhoods to multi-syllable words. Each vowel group is considered within the context of its rime, with each syllable considered separately. Consonant neighborhoods consist of orthographic clusters which correspond to the same location in the word. This results in 4 consonant cluster locations: first syllable onset, first syllable coda, second syllable onset, and second syllable coda. Consonant cluster neighborhoods include the preceeding vowel for coda consonants, and the following vowel for onset consonants.

The notion of exception words is also not universally agreed upon. Precisely which words are exceptions is a function of the working definition of pronunciation and regularity for the experiment at hand. Given a definition of neighborhood, then, exception words can be defined as those words which do not agree with the phonological mapping favored by the majority of items in that particular neighborhood. Alternatively, in cases assuming a set of rules for grapheme-phoneme correspondence, exception words are those which violate the rules which define the majority of pronunciations. For this investigation, single syllable exception words are those defined as exception by the T&M neighborhood definition. For instance, PINT would be considered an exception word compared to its neighbors MINT, TINT, HINT, etc. Coltheart, on the other hand, defines exception words to be those for which his GPC rules produce incorrect pronunciation. Since we are concerned with addressing Coltheart's claims, these 2-syllable exception words will also be used here.

## 4.1  Results

**Single syllable words**  For each phoneme position, we compare each word with irregularity at that position with its neighbors, counting the number of enemies (words with alternate pronunciation at the supposed irregularity) and friends (words with pronunciation in agreement) that it has. The T&M neighborhood numbers (words containing the same vowel grouping and final consonant cluster) used in Figure 1 are found in (Taraban and McClelland, 1987). For each word, we calculate its (enemy) / (friend+enemy) ratio; these ratios are then averaged over all the words in the position set. The results using neighborhoods as defined in Taraban & McClelland clearly show the desired rank ordering of effect. First-position-irregularity words have more "enemies" and fewer "friends" than third-position-irregularity words, with the second-position words falling in the middle as desired. We suggest that this statistical regularity in the data is what the above networks capture.

However convincing these results may be, they do not fully address Coltheart's data, which is for two syllable words of five phonemes or phoneme clusters, with irregularities at each of five possible positions. Also, due to the size of the T&M data set, there are only 2 members in the position 1 set, and the single-syllable data only goes up to phoneme position 3. The neighborhoods for the two-syllable data set were thus examined.

**Two syllable results**  Recall that the two-syllable test words are those used in the (Coltheart and Rastle, 1994) subject study, for which naming latency differences are shown in Table 1. Coltheart's 1-letter-different neighborhood definition

is not very informative in this case, since by this criterion most of the target words provided in (Coltheart and Rastle, 1994) are loners (i.e., have no neighbors at all). However, using a neighborhood based on T&M-2 recreates the desired ranking (Figure 2) as indicated by the ratio of hindering pronunciations to the total of the helping and hindering pronunciations. As with the single syllable words, each test word is compared with its neighbor words and the (enemy)/(friend+enemy) ratio is calculated. Averaging over the words in each position set, we again see that words with early irregularities are at a support disadvantage compared to words with late irregularities.

## 5 Summary

Dual-Route models claim the irregularity position effect can only be accounted for by two-route models with left-to-right activation of phonemes, and interaction between GPC rules and the lexicon. The work presented in this paper refutes this claim by presenting results from feed-forward connectionist networks which show the same rank ordering of latency. Further, an analysis of orthographic neighborhoods shows *why* the networks can do this: the effect is based on a statistical interaction between friend/enemy support and position. Words with irregular orthographic-phonemic correspondence at word beginning have less support from their neighbors than words with later irregularities; it is this difference which explains the latency results. The resulting statistical regularity is then easily captured by connectionist networks exposed to representative data sets.

## Footnotes

[1] Friends are words with the same pronunciations for the ambiguous letter-to-sound correspondence; enemies are words with different pronunciations.

[2] Available via ftp://ftp.cs.cmu.edu/project/fgdata/dict/

## References

Coltheart, M., Curitis, B., Atkins, P., and Haller, M. (1993). Models of reading aloud: Dual-route and parallel-distributed-processing approaches. *Psychological Review*, 100(4):589–608.

Coltheart, M. and Rastle, K. (1994). Serial processing in reading aloud: Evidence for dual route models of reading. *Journal of Experimental Psychology: Human Perception and Performance*, 20(6):1197–1211.

Kucera, H. and Francis, W. (1967). *Computational Analysis of Present-Day American English*. Brown University Press, Providence, RI.

McClelland, J. and Rumelhart, D. (1981). An interactive activation model of context effects in letter perception: Part 1. an account of basic findings. *Psychological Review*, 88:375–407.

Peereman, R. (1995). Naming regular and exception words: Further examination of the effect of phonological dissension among lexical neighbours. *European Journal of Cognitive Psychology*, 7(3):307–330.

Plaut, D., McClelland, J., Seidenberg, M., and Patterson, K. (1996). Understanding normal and impaired word reading: Computational principles in quasi-regular domains. *Psychological Review*, 103(1):56–115.

Seidenberg, M. and McClelland, J. (1989). A distributed, developmental model of word recognition and naming. *Psychological Review*, 96:523–568.

Taraban, R. and McClelland, J. (1987). Conspiracy effects in word pronunciation. *Journal of Memory and Language*, 26:608–631.

Treiman, R. and Chafetz, J. (1987). Are there onset- and rime-like units in printed words? In Coltheart, M., editor, *Attention and Performance XII: The Psychology of Reading*. Erlbaum, Hillsdale, NJ.
